# Adaptor Grammars: A Framework for Specifying Compositional Nonparametric Bayesian Models

**Mark Johnson**
Microsoft Research / Brown University
Mark_Johnson@Brown.edu

**Thomas L. Griffiths**
University of California, Berkeley
Tom_Griffiths@Berkeley.edu

**Sharon Goldwater**
Stanford University
sgwater@gmail.com

## Abstract

This paper introduces adaptor grammars, a class of probabilistic models of language that generalize probabilistic context-free grammars (PCFGs). Adaptor grammars augment the probabilistic rules of PCFGs with "adaptors" that can induce dependencies among successive uses. With a particular choice of adaptor, based on the Pitman-Yor process, nonparametric Bayesian models of language using Dirichlet processes and hierarchical Dirichlet processes can be written as simple grammars. We present a general-purpose inference algorithm for adaptor grammars, making it easy to define and use such models, and illustrate how several existing nonparametric Bayesian models can be expressed within this framework.

## 1 Introduction

Probabilistic models of language make two kinds of substantive assumptions: assumptions about the structures that underlie language, and assumptions about the probabilistic dependencies in the process by which those structures are generated. Typically, these assumptions are tightly coupled. For example, in probabilistic context-free grammars (PCFGs), structures are built up by applying a sequence of context-free rewrite rules, where each rule in the sequence is selected independently at random. In this paper, we introduce a class of probabilistic models that weaken the independence assumptions made in PCFGs, which we call *adaptor grammars*. Adaptor grammars insert additional stochastic processes called *adaptors* into the procedure for generating structures, allowing the expansion of a symbol to depend on the way in which that symbol has been rewritten in the past. Introducing dependencies among the applications of rewrite rules extends the set of distributions over linguistic structures that can be characterized by a simple grammar.

Adaptor grammars provide a simple framework for defining nonparametric Bayesian models of language. With a particular choice of adaptor, based on the Pitman-Yor process [1, 2, 3], simple context-free grammars specify distributions commonly used in nonparametric Bayesian statistics, such as Dirichlet processes [4] and hierarchical Dirichlet processes [5]. As a consequence, many nonparametric Bayesian models that have been used in computational linguistics, such as models of morphology [6] and word segmentation [7], can be expressed as adaptor grammars. We introduce a general-purpose inference algorithm for adaptor grammars, which makes it easy to define nonparametric Bayesian models that generate different linguistic structures and perform inference in those models.

The rest of this paper is structured as follows. Section 2 introduces the key technical ideas we will use. Section 3 defines adaptor grammars, while Section 4 presents some examples. Section 5 describes the Markov chain Monte Carlo algorithm we have developed to sample from the posterior

distribution over structures generated by an adaptor grammar. Software implementing this algorithm is available from http://cog.brown.edu/~mj/Software.htm.

## 2 Background

In this section, we introduce the two technical ideas that are combined in the adaptor grammars discussed here: probabilistic context-free grammars, and the Pitman-Yor process. We adopt a non-standard formulation of PCFGs in order to emphasize that they are a kind of recursive mixture, and to establish the formal devices we use to specify adaptor grammars.

### 2.1 Probabilistic context-free grammars

A *context-free grammar* (CFG) is a quadruple $(N, W, R, S)$ where $N$ is a finite set of *nonterminal symbols*, $W$ is a finite set of *terminal symbols* disjoint from $N$, $R$ is a finite set of productions or *rules* of the form $A \rightarrow \beta$ where $A \in N$ and $\beta \in (N \cup W)^\star$ (the Kleene closure of the terminal and nonterminal symbols), and $S \in N$ is a distinguished nonterminal called the *start symbol*. A CFG associates with each symbol $A \in N \cup W$ a set $\mathcal{T}_A$ of finite, labeled, ordered trees. If $A$ is a terminal symbol then $\mathcal{T}_A$ is the singleton set consisting of a unit tree (i.e., containing a single node) labeled $A$. The sets of trees associated with nonterminals are defined recursively as follows:

$$\mathcal{T}_A = \bigcup_{A \rightarrow B_1 \dots B_n \in R_A} \text{TREE}_A(\mathcal{T}_{B_1}, \dots, \mathcal{T}_{B_n})$$

where $R_A$ is the subset of productions in $R$ with left-hand side $A$, and $\text{TREE}_A(\mathcal{T}_{B_1}, \dots, \mathcal{T}_{B_n})$ is the set of all trees whose root node is labeled $A$, that have $n$ immediate subtrees, and where the $i$th subtree is a member of $\mathcal{T}_{B_i}$. The set of trees generated by the CFG is $\mathcal{T}_S$, and the language generated by the CFG is the set $\{\text{YIELD}(t) : t \in \mathcal{T}_S\}$ of terminal strings or yields of the trees $\mathcal{T}_S$.

A *probabilistic context-free grammar* (PCFG) is a quintuple $(N, W, R, S, \theta)$, where $(N, W, R, S)$ is a CFG and $\theta$ is a vector of non-negative real numbers indexed by productions $R$ such that

$$\sum_{A \rightarrow \beta \in R_A} \theta_{A \rightarrow \beta} = 1.$$

Informally, $\theta_{A \rightarrow \beta}$ is the probability of expanding the nonterminal $A$ using the production $A \rightarrow \beta$. $\theta$ is used to define a distribution $G_A$ over the trees $\mathcal{T}_A$ for each symbol $A$. If $A$ is a terminal symbol, then $G_A$ is the distribution that puts all of its mass on the unit tree labeled $A$. The distributions $G_A$ for nonterminal symbols are defined recursively over $\mathcal{T}_A$ as follows:

$$G_A = \sum_{A \rightarrow B_1 \dots B_n \in R_A} \theta_{A \rightarrow B_1 \dots B_n} \text{TREEDIST}_A(G_{B_1}, \dots, G_{B_n}) \tag{1}$$

where $\text{TREEDIST}_A(G_{B_1}, \dots, G_{B_n})$ is the distribution over $\text{TREE}_A(\mathcal{T}_{B_1}, \dots, \mathcal{T}_{B_n})$ satisfying:

$$\text{TREEDIST}_A(G_1, \dots, G_n) \left( \overset{A}{\overbrace{t_1 \ \dots \ t_n}} \right) = \prod_{i=1}^{n} G_i(t_i).$$

That is, $\text{TREEDIST}_A(G_1, \dots, G_n)$ is a distribution over trees where the root node is labeled $A$ and each subtree $t_i$ is generated independently from $G_i$; it is this assumption that adaptor grammars relax. The distribution over trees generated by the PCFG is $G_S$, and the probability of a string is the sum of the probabilities of all trees with that string as their yields.

### 2.2 The Pitman-Yor process

The Pitman-Yor process [1, 2, 3] is a stochastic process that generates partitions of integers. It is most intuitively described using the metaphor of seating customers at a restaurant. Assume we have a numbered sequence of tables, and $z_i$ indicates the number of the table at which the $i$th customer is seated. Customers enter the restaurant sequentially. The first customer sits at the first table, $z_1 = 1$, and the $n + 1$st customer chooses a table from the distribution

$$z_{n+1} | z_1, \dots, z_n \sim \frac{ma + b}{n + b} \delta_{m+1} + \sum_{k=1}^{m} \frac{n_k - a}{n + b} \delta_k \tag{2}$$

where $m$ is the number of different indices appearing in the sequence $\mathbf{z} = (z_1, \ldots, z_n)$, $n_k$ is the number of times $k$ appears in $\mathbf{z}$, and $\delta_k$ is the Kronecker delta function, i.e., the distribution that puts all of its mass on $k$. The process is specified by two real-valued parameters, $a \in [0, 1]$ and $b \geq 0$. The probability of a particular sequence of assignments, $\mathbf{z}$, with a corresponding vector of table counts $\mathbf{n} = (n_1, \ldots, n_m)$ is

$$\mathrm{P}(\mathbf{z}) \;=\; \mathrm{PY}(\mathbf{n} \,|\, a, b) \;=\; \frac{\prod_{k=1}^{m}(a(k-1)+b)\,\prod_{j=1}^{n_k-1}(j-a)}{\prod_{i=0}^{n-1}(i+b)}. \tag{3}$$

From this it is easy to see that the distribution produced by the Pitman-Yor process is *exchangeable*, with the probability of $\mathbf{z}$ being unaffected by permutation of the indices of the $z_i$.

Equation 2 instantiates a kind of "rich get richer" dynamics, with customers being more likely to sit at more popular tables. We can use the Pitman-Yor process to define distributions with this character on any desired domain. Assume that every table in our restaurant has a value $x_j$ placed on it, with those values being generated from an exchangeable distribution $G$, which we will refer to as the *generator*. Then, we can sample a sequence of variables $\mathbf{y} = (y_1, \ldots, y_n)$ by using the Pitman-Yor process to produce $\mathbf{z}$ and setting $y_i = x_{z_i}$. Intuitively, this corresponds to customers entering the restaurant, and emitting the values of the tables they choose. The distribution defined on $\mathbf{y}$ by this process will be exchangeable, and has two interesting special cases that depend on the parameters of the Pitman-Yor process. When $a = 1$, every customer is assigned to a new table, and the $y_i$ are drawn from $G$. When $a = 0$, the distribution on the $y_i$ is that induced by the Dirichlet process [4], a stochastic process that is commonly used in nonparametric Bayesian statistics, with concentration parameter $b$ and base distribution $G$.

We can also identify another scheme that generates the distribution outlined in the previous paragraph. Let $H$ be a discrete distribution produced by generating a set of atoms $\mathbf{x}$ from $G$ and weights on those atoms from the two-parameter Poisson-Dirichlet distribution [2]. We could then generate a sequence of samples $\mathbf{y}$ from $H$. If we integrate over values of $H$, the distribution on $\mathbf{y}$ is the same as that obtained via the Pitman-Yor process [2, 3].

## 3 Adaptor grammars

In this section, we use the ideas introduced in the previous section to give a formal definition of adaptor grammars. We first state this definition in full generality, allowing any choice of adaptor, and then consider the case where the adaptor is based on the Pitman-Yor process in more detail.

### 3.1 A general definition of adaptor grammars

Adaptor grammars extend PCFGs by inserting an additional component called an *adaptor* into the PCFG recursion (Equation 1). An adaptor $C$ is a function from a distribution $G$ to a distribution over distributions with the same support as $G$. An *adaptor grammar* is a sextuple $(N, W, R, S, \theta, \mathbf{C})$ where $(N, W, R, S, \theta)$ is a PCFG and the adaptor vector $\mathbf{C}$ is a vector of (parameters specifying) adaptors indexed by $N$. That is, $C_A$ maps a distribution over trees $\mathcal{T}_A$ to another distribution over $\mathcal{T}_A$, for each $A \in N$. An adaptor grammar associates each symbol with two distributions $G_A$ and $H_A$ over $\mathcal{T}_A$. If $A$ is a terminal symbol then $G_A$ and $H_A$ are distributions that put all their mass on the unit tree labeled $A$, while $G_A$ and $H_A$ for nonterminal symbols are defined as follows:[1]

$$G_A \;=\; \sum_{A \to B_1 \ldots B_n \in R_A} \theta_{A \to B_1 \ldots B_n} \textsc{TreeDist}_A(H_{B_1}, \ldots, G_{H_n}) \tag{4}$$

$$H_A \;\sim\; C_A(G_A)$$

The intuition here is that $G_A$ instantiates the PCFG recursion, while the introduction of $H_A$ makes it possible to modify the independence assumptions behind the resulting distribution through the choice of the adaptor, $C_A$. If the adaptor is the identity function, with $H_A = G_A$, the result is just a PCFG. However, other distributions over trees can be defined by choosing other adaptors. In practice, we integrate over $H_A$, to define a single distribution on trees for any choice of adaptors $\mathbf{C}$.

## 3.2 Pitman-Yor adaptor grammars

The definition given above allows the adaptors to be any appropriate process, but our focus in the remainder of the paper will be on the case where the adaptor is based on the Pitman-Yor process. Pitman-Yor processes can cache, i.e., increase the probability of, frequently occurring trees. The capacity to replace the independent selection of rewrite rules with an exchangeable stochastic process enables adaptor grammars based on the Pitman-Yor process to define probability distributions over trees that cannot be expressed using PCFGs.

A *Pitman-Yor adaptor grammar* (PYAG) is an adaptor grammar where the adaptors $\mathbf{C}$ are based on the Pitman-Yor process. A Pitman-Yor adaptor $C_A(G_A)$ is the distribution obtained by generating a set of atoms from the distribution $G_A$ and weights on those atoms from the two-parameter Poisson-Dirichlet distribution. A PYAG has an adaptor $C_A$ with parameters $a_A$ and $b_A$ for each non-terminal $A \in N$. As noted above, if $a_A = 1$ then the Pitman-Yor process is the identity function, so $A$ is expanded in the standard manner for a PCFG. Each adaptor $C_A$ will also be associated with two vectors, $\mathbf{x}_A$ and $\mathbf{n}_A$, that are needed to compute the probability distribution over trees. $\mathbf{x}_A$ is the sequence of previously generated subtrees with root nodes labeled $A$. Having been "cached" by the grammar, these now have higher probability than other subtrees. $\mathbf{n}_A$ lists the counts associated with the subtrees in $\mathbf{x}_A$. The adaptor state can thus be summarized as $C_A = (a_A, b_A, \mathbf{x}_A, \mathbf{n}_A)$.

A *Pitman-Yor adaptor grammar analysis* $u = (t, \ell)$ is a pair consisting of a parse tree $t \in \mathcal{T}_S$ together with an index function $\ell(\cdot)$. If $q$ is a nonterminal node in $t$ labeled $A$, then $\ell(q)$ gives the index of the entry in $\mathbf{x}_A$ for the subtree $t'$ of $t$ rooted at $q$, i.e., such that $\mathbf{x}_{A_{\ell(q)}} = t'$. The sequence of analyses $\mathbf{u} = (u_1, \ldots, u_n)$ generated by an adaptor grammar contains sufficient information to compute the adaptor state $\mathbf{C}(\mathbf{u})$ after generating $\mathbf{u}$: the elements of $\mathbf{x}_A$ are the distinctly indexed subtrees of $\mathbf{u}$ with root label $A$, and their frequencies $\mathbf{n}_A$ can be found by performing a top-down traversal of each analysis in turn, only visiting the children of a node $q$ when the subanalysis rooted at $q$ is encountered for the first time (i.e., when it is added to $\mathbf{x}_A$).

# 4 Examples of Pitman-Yor adaptor grammars

Pitman-Yor adaptor grammars provide a framework in which it is easy to define compositional non-parametric Bayesian models. The use of adaptors based on the Pitman-Yor process allows us to specify grammars that correspond to Dirichlet processes [4] and hierarchical Dirichlet processes [5]. Once expressed in this framework, a general-purpose inference algorithm can be used to calculate the posterior distribution over analyses produced by a model. In this section, we illustrate how existing nonparametric Bayesian models used for word segmentation [7] and morphological analysis [6] can be expressed as adaptor grammars, and describe the results of applying our inference algorithm in these models. We postpone the presentation of the algorithm itself until Section 5.

## 4.1 Dirichlet processes and word segmentation

Adaptor grammars can be used to define Dirichlet processes with discrete base distributions. It is straightforward to write down an adaptor grammar that defines a Dirichlet process over all strings:

$$\begin{aligned} \text{Word} &\rightarrow \text{Chars} \\ \text{Chars} &\rightarrow \text{Char} \\ \text{Chars} &\rightarrow \text{Chars Char} \end{aligned} \tag{5}$$

The productions expanding Char to all possible characters are omitted to save space. The start symbol for this grammar is Word. The parameters $a_{\text{Char}}$ and $a_{\text{Chars}}$ are set to 1, so the adaptors for Char and Chars are the identity function and $H_{\text{Chars}} = G_{\text{Chars}}$ is the distribution over words produced by sampling each character independently (i.e., a "monkeys at typewriters" model). Finally, $a_{\text{Word}}$ is set to 0, so the adaptor for Word is a Dirichlet process with concentration parameter $b_{\text{Word}}$.

This grammar generates all possible strings of characters and assigns them simple right-branching structures of no particular interest, but the Word adaptor changes their distribution to one that reflects the frequencies of previously generated words. Initially, the Word adaptor is empty (i.e., $\mathbf{x}_{\text{Word}}$ is empty), so the first word $s_1$ generated by the grammar is distributed according to $G_{\text{Chars}}$. However, the second word can be generated in two ways: either it is retrieved from the adaptor's cache (and

hence is $s_1$) with probability $1/(1 + b_{\text{Word}})$, or else with probability $b_{\text{Word}}/(1 + b_{\text{Word}})$ it is a new word generated by $G_{\text{Chars}}$. After $n$ words have been emitted, Word puts mass $n/(n + b_{\text{Word}})$ on those words and reserves mass $b_{\text{Word}}/(n + b_{\text{Word}})$ for new words (i.e., generated by Chars).

We can extend this grammar to a simple unigram word segmentation model by adding the following productions, changing the start label to Words and setting $a_{\text{Words}} = 1$.

$$
\begin{aligned}
\text{Words} &\rightarrow \text{Word} \\
\text{Words} &\rightarrow \text{Word Words}
\end{aligned}
$$

This grammar generates sequences of Word subtrees, so it implicitly segments strings of terminals into a sequence of words, and in fact implements the word segmentation model of [7]. We applied the grammar above with the algorithm described in Section 5 to a corpus of unsegmented child-directed speech [8]. The input strings are sequences of phonemes such as *WAtIzIt*. A typical parse might consist of Words dominating three Word subtrees, each in turn dominating the phoneme sequences *Wat*, *Iz* and *It* respectively. Using the sampling procedure described in Section 5 with $b_{\text{Word}} = 30$, we obtained a segmentation which identified words in unsegmented input with 0.64 precision, 0.51 recall, and 0.56 f-score, which is consistent with the results presented for the unigram model of [7] on the same data.

## 4.2 Hierarchical Dirichlet processes and morphological analysis

An adaptor grammar with more than one adapted nonterminal can implement a hierarchical Dirichlet process. A hierarchical Dirichlet process that uses the Word process as a generator can be defined by adding the production Word1 $\rightarrow$ Word to (5) and making Word1 the start symbol. Informally, Word1 generates words either from its own cache $\mathbf{x}_{\text{Word1}}$ or from the Word distribution. Word itself generates words either from $\mathbf{x}_{\text{Word}}$ or from the "monkeys at typewriters" model Chars.

A slightly more elaborate grammar can implement the morphological analysis described in [6]. Words are analysed into stem and suffix substrings; e.g., the word *jumping* is analysed as a stem *jump* and a suffix *ing*. As [6] notes, one of the difficulties in constructing a probabilistic account of such suffixation is that the relative frequencies of suffixes varies dramatically depending on the stem. That paper used a Pitman-Yor process to effectively dampen this frequency variation, and the adaptor grammar described here does exactly the same thing. The productions of the adaptor grammar are as follows, where Chars is "monkeys at typewriters" once again:

$$
\begin{aligned}
\text{Word} &\rightarrow \text{Stem Suffix} \\
\text{Word} &\rightarrow \text{Stem} \\
\text{Stem} &\rightarrow \text{Chars} \\
\text{Suffix} &\rightarrow \text{Chars}
\end{aligned}
$$

We now give an informal description of how samples might be generated by this grammar. The nonterminals Word, Stem and Suffix are associated with Pitman-Yor adaptors. Stems and suffixes that occur in many words are associated with highly probable cache entries, and so have much higher probability than under the Chars PCFG subgrammar.

Figure 1 depicts a possible state of the adaptors in this adaptor grammar after generating the three words *walking*, *jumping* and *walked*. Such a state could be generated as follows. Before any strings are generated all of the adaptors are empty. To generate the first word we must sample from $H_{\text{Word}}$, as there are no entries in the Word adaptor. Sampling from $H_{\text{Word}}$ requires sampling from $G_{\text{Stem}}$ and perhaps also $G_{\text{Suffix}}$, and eventually from the Chars distributions. Supposing that these return *walk* and *ing* as Stem and Suffix strings respectively, the adaptor entries after generating the first word *walking* consist of the first entries for Word, Stem and Suffix.

In order to generate another Word we first decide whether to select an existing word from the adaptor, or whether to generate the word using $G_{\text{Word}}$. Suppose we choose the latter. Then we must sample from $H_{\text{Stem}}$ and perhaps also from $H_{\text{Suffix}}$. Suppose we choose to generate the new stem *jump* from $G_{\text{Stem}}$ (resulting in the second entry in the Stem adaptor) but choose to reuse the existing Suffix adaptor entry, resulting in the word *jumping*. The third word *walked* is generated in a similar fashion: this time the stem is the first entry in the Stem adaptor, but the suffix *ed* is generated from $G_{\text{Suffix}}$ and becomes the second entry in the Suffix adaptor.

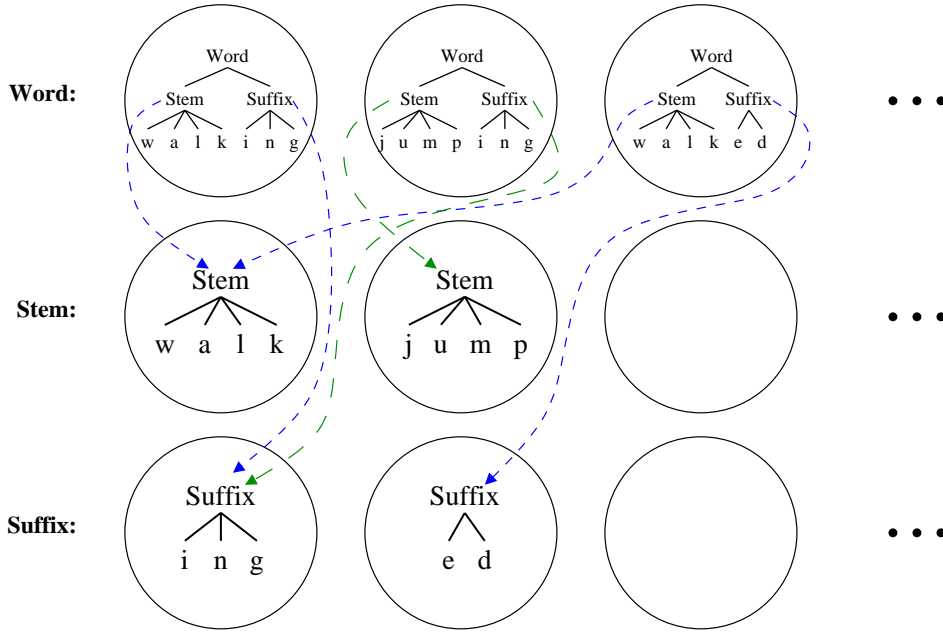

Figure 1: A depiction of a possible state of the Pitman-Yor adaptors in the adaptor grammar of Section 4.2 after generating *walking*, *jumping* and *walked*.

The model described in [6] is more complex than the one just described because it uses a hidden "morphological class" variable that determines which stem-suffix pair is selected. The morphological class variable is intended to capture morphological variation; e.g., the present continuous form *skipping* is formed by suffixing *ping* instead of the *ing* form using in *walking* and *jumping*. This can be expressed using an adaptor grammar with productions that instantiate the following schema:

$$
\begin{array}{llll}
\text{Word} & \rightarrow & \text{Word}_c & \qquad \text{Stem}_c \rightarrow \text{Chars} \\
\text{Word}_c & \rightarrow & \text{Stem}_c\, \text{Suffix}_c & \qquad \text{Suffix}_c \rightarrow \text{Chars} \\
\text{Word}_c & \rightarrow & \text{Stem}_c &
\end{array}
$$

Here $c$ ranges over the hidden morphological classes, and the productions expanding Chars and Char are as before. We set the adaptor parameter $a_{\text{Word}} = 1$ for the start nonterminal symbol Word, so we adapt the $\text{Word}_c$, $\text{Stem}_c$ and $\text{Suffix}_c$ nonterminals for each hidden class $c$.

Following [6], we used this grammar with six hidden classes $c$ to segment 170,015 orthographic verb tokens from the Penn Wall Street Journal corpus, and set $a = 0$ and $b = 500$ for the adapted nonterminals. Although we trained on all verbs in the corpus, we evaluated the segmentation produced by the inference procedure described below on just the verbs whose infinitival stems were a prefix of the verb itself (i.e., we evaluated *skipping* but ignored *wrote*, since its stem *write* is not a prefix). Of the 116,129 tokens we evaluated, 70% were correctly segmented, and of the 7,170 verb types, 66% were correctly segmented. Many of the errors were in fact linguistically plausible: e.g., *eased* was analysed as a stem *eas* followed by a suffix *ed*, permitting the grammar to also generate *easing* as *eas* plus *ing*.

## 5 Bayesian inference for Pitman-Yor adaptor grammars

The results presented in the previous section were obtained by using a Markov chain Monte Carlo (MCMC) algorithm to sample from the posterior distribution over PYAG analyses $\mathbf{u} = (u_1, \ldots, u_n)$ given strings $\mathbf{s} = (s_1, \ldots, s_n)$, where $s_i \in W^\star$ and $u_i$ is the analysis of $s_i$. We assume we are given a CFG $(N, W, R, S)$, vectors of Pitman-Yor adaptor parameters $\mathbf{a}$ and $\mathbf{b}$, and a Dirichlet prior with hyperparameters $\alpha$ over production probabilities $\theta$, i.e.:

$$
\mathrm{P}(\theta \,|\, \alpha) = \prod_{A \in N} \frac{1}{B(\alpha_A)} \prod_{A \rightarrow \beta \in R_A} \theta_{A \rightarrow \beta}^{\,\alpha_{A \rightarrow \beta} - 1} \quad \text{where:}
$$

$$B(\alpha_A) = \frac{\prod_{A \to \beta \in R_A} \Gamma(\alpha_{A \to \beta})}{\Gamma(\sum_{A \to \beta \in R_A} \alpha_{A \to \beta})}$$

with $\Gamma(x)$ being the generalized factorial function, and $\alpha_A$ is the subsequence of $\alpha$ indexed by $R_A$ (i.e., corresponding to productions that expand $A$). The joint probability of $\mathbf{u}$ under this PYAG, integrating over the distributions $H_A$ generated from the two-parameter Poisson-Dirichlet distribution associated with each adaptor, is

$$P(\mathbf{u} \,|\, \alpha, \mathbf{a}, \mathbf{b}) = \prod_{A \in N} \frac{B(\alpha_A + \mathbf{f}_A(\mathbf{x}_A))}{B(\alpha_A)} PY(\mathbf{n}_A(\mathbf{u}) | \mathbf{a}, \mathbf{b}) \tag{6}$$

where $f_{A \to \beta}(\mathbf{x}_A)$ is the number of times the root node of a tree in $\mathbf{x}_A$ is expanded by production $A \to \beta$, and $\mathbf{f}_A(\mathbf{x}_A)$ is the sequence of such counts (indexed by $r \in R_A$). Informally, the first term in (6) is the probability of generating the topmost node in each analysis in adaptor $C_A$ (the rest of the tree is generated by another adaptor), while the second term (from Equation 3) is the probability of generating a Pitman-Yor adaptor with counts $\mathbf{n}_A$.

The posterior distribution over analyses $\mathbf{u}$ given strings $\mathbf{s}$ is obtained by normalizing $P(\mathbf{u} \,|\, \alpha, \mathbf{a}, \mathbf{b})$ over all analyses $\mathbf{u}$ that have $\mathbf{s}$ as their yield. Unfortunately, computing this distribution is intractable. Instead, we draw samples from this distribution using a component-wise Metropolis-Hastings sampler, proposing changes to the analysis $u_i$ for each string $s_i$ in turn. The proposal distribution is constructed to approximate the conditional distribution over $u_i$ given $s_i$ and the analyses of all other strings $\mathbf{u}_{-i}$, $P(u_i|s_i, \mathbf{u}_{-i})$. Since there does not seem to be an efficient (dynamic programming) algorithm for directly sampling from $P(u_i|s_i, \mathbf{u}_{-i})$,[2] we construct a PCFG $G'(\mathbf{u}_{-i})$ on the fly whose parse trees can be transformed into PYAG analyses, and use this as our proposal distribution.

## 5.1 The PCFG approximation $G'(\mathbf{u}_{-i})$

A PYAG can be viewed as a special kind of PCFG which adapts its production probabilities depending on its history. The PCFG approximation $G'(\mathbf{u}_{-i}) = (N, W, R', S, \theta')$ is a static snapshot of the adaptor grammar given the sentences $\mathbf{s}_{-i}$ (i.e., all of the sentences in $\mathbf{s}$ except $s_i$). Given an adaptor grammar $H = (N, W, R, S, \mathbf{C})$, let:

$$R' = R \cup \bigcup_{A \in N} \{A \to \text{YIELD}(x) : x \in \mathbf{x}_A\}$$

$$\theta'_{A \to \beta} = \left(\frac{m_A a_A + b_A}{n_A + b_A}\right) \left(\frac{f_{A \to \beta}(\mathbf{x}_A) + \alpha_{A \to \beta}}{m_A + \sum_{A \to \beta \in R_A} \alpha_{A \to \beta}}\right) + \sum_{k : \text{YIELD}(X_{A_k}) = \beta} \left(\frac{n_{A_k} - a_A}{n_A + b_A}\right)$$

where $\text{YIELD}(x)$ is the terminal string or yield of the tree $x$ and $m_A$ is the length of $\mathbf{x}_A$. $R'$ contains all of the productions $R$, together with productions representing the adaptor entries $\mathbf{x}_A$ for each $A \in N$. These additional productions rewrite directly to strings of terminal symbols, and their probability is the probability of the adaptor $C_A$ generating the corresponding value $x_{A_k}$.

The two terms to the left of the summation specify the probability of selecting a production from the original productions $R$. The first term is the probability of adaptor $C_A$ generating a new value, and the second term is the MAP estimate of the production's probability, estimated from the root expansions of the trees $\mathbf{x}_A$.

It is straightforward to map parses of a string $s$ produced by $G'$ to corresponding adaptor analyses for the adaptor grammar $H$ (it is possible for a single production of $R'$ to correspond to several adaptor entries so this mapping may be non-deterministic). This means that we can use the PCFG $G'$ with an efficient PCFG sampling procedure [9] to generate possible adaptor grammar analyses for $u_i$.

## 5.2 A Metropolis-Hastings algorithm

The previous section described how to sample adaptor analyses $u$ for a string $s$ from a PCFG approximation $G'$ to an adaptor grammar $H$. We use this as our proposal distribution in a Metropolis-

Hastings algorithm. If $u_i$ is the current analysis of $s_i$ and $u'_i \neq u_i$ is a proposal analysis sampled from $P(U_i|s_i, G'(\mathbf{u}_{-i}))$ we accept the proposal $u_i$ with probability $A(u_i, u'_i)$, where:

$$A(u_i, u'_i) \;\;=\;\; \min\left\{1, \frac{P(\mathbf{u}' \,|\, \alpha, \mathbf{a}, \mathbf{b})\, P(u_i \,|\, s_i, G'(\mathbf{u}_{-i}))}{P(\mathbf{u} \,|\, \alpha, \mathbf{a}, \mathbf{b})\, P(u'_i \,|\, s_i, G'(\mathbf{u}_{-i}))}\right\}$$

where $\mathbf{u}'$ is the same as $\mathbf{u}$ except that $u'_i$ replaces $u_i$. Except when the number of training strings $\mathbf{s}$ is very small, we find that only a tiny fraction (less than $1\%$) of proposals are rejected, presumably because the probability of an adaptor analysis does not change significantly within a single string.

Our inference procedure is as follows. Given a set of training strings $\mathbf{s}$ we choose an initial set of analyses for them at random. At each iteration we pick a string $s_i$ from $\mathbf{s}$ at random, and sample a parse for $s_i$ from the PCFG approximation $G'(\mathbf{u}_{-i})$, updating $\mathbf{u}$ when the Metropolis-Hastings procedure accepts the proposed analysis. At convergence the $\mathbf{u}$ produced by this procedure are samples from the posterior distribution over analyses given $\mathbf{s}$, and samples from the posterior distribution over adaptor states $\mathbf{C}(\mathbf{u})$ and production probabilities $\theta$ can be computed from them.

## 6 Conclusion

The strong independence assumptions of probabilistic context-free grammars tightly couple compositional structure with the probabilistic generative process that produces that structure. Adaptor grammars relax that coupling by inserting an additional stochastic component into the generative process. Pitman-Yor adaptor grammars use adaptors based on the Pitman-Yor process. This choice makes it possible to express Dirichlet process and hierarchical Dirichlet process models over discrete domains as simple context-free grammars. We have proposed a general-purpose inference algorithm for adaptor grammars, which can be used to sample from the posterior distribution over analyses produced by any adaptor grammar. While our focus here has been on demonstrating that this algorithm can be used to produce equivalent results to existing nonparametric Bayesian models used for word segmentation and morphological analysis, the great promise of this framework lies in its simplification of specifying and using such models, providing a basic toolbox that will facilitate the construction of more sophisticated models.

### Acknowledgments

This work was performed while all authors were at the Cognitive and Linguistic Sciences Department at Brown University and supported by the following grants: NIH R01-MH60922 and RO1-DC000314, NSF 9870676, 0631518 and 0631667, the DARPA CALO project and DARPA GALE contract HR0011-06-2-0001.

## Footnotes

[1]This definition allows an adaptor grammar to include self-recursive or mutually recursive CFG productions (e.g., $X \to X\,Y$ or $X \to Y\,Z, Y \to X\,W$). Such recursion complicates inference, so we restrict ourselves to grammars where the adapted nonterminals are not recursive.

[2]The independence assumptions of PCFGs play an important role in making dynamic programming possible. In PYAGs, the probability of a subtree adapts dynamically depending on the other subtrees in $\mathbf{u}$, including those in $u_i$.

## References

[1] J. Pitman. Exchangeable and partially exchangeable random partitions. *Probability Theory and Related Fields*, 102:145–158, 1995.

[2] J. Pitman and M. Yor. The two-parameter Poisson-Dirichlet distribution derived from a stable subordinator. *Annals of Probability*, 25:855–900, 1997.

[3] H. Ishwaran and L. F. James. Generalized weighted Chinese restaurant processes for species sampling mixture models. *Statistica Sinica*, 13:1211–1235, 2003.

[4] T. Ferguson. A Bayesian analysis of some nonparametric problems. *The Annals of Statistics*, 1:209–230, 1973.

[5] Y. W. Teh, M. Jordan, M. Beal, and D. Blei. Hierarchical Dirichlet processes. *Journal of the American Statistical Association*, to appear.

[6] S. Goldwater, T. L. Griffiths, and M. Johnson. Interpolating between types and tokens by estimating power-law generators. In *Advances in Neural Information Processing Systems 18*, 2006.

[7] S. Goldwater, T. L. Griffiths, and M. Johnson. Contextual dependencies in unsupervised word segmentation. In *Proceedings of the 44th Annual Meeting of the Association for Computational Linguistics*, 2006.

[8] M. Brent. An efficient, probabilistically sound algorithm for segmentation and word discovery. *Machine Learning*, 34:71–105, 1999.

[9] J. Goodman. *Parsing inside-out*. PhD thesis, Harvard University, 1998. available from http://research.microsoft.com/~joshuago/.
